# A Convex Upper Bound on the Log-Partition Function for Binary Graphical Models

**Laurent El Ghaoui**
Department of Electrical Engineering and Computer Science
University of California Berkeley
Berkeley, CA 9470
elghaoui@eecs.berkeley.edu

**Assane Gueye**
Department of Electrical Engineering and Computer Science
University of California Berkeley
Berkeley, CA 9470
agueye@eecs.berkeley.edu

## Abstract

We consider the problem of bounding from above the log-partition function corresponding to second-order Ising models for binary distributions. We introduce a new bound, the cardinality bound, which can be computed via convex optimization. The corresponding error on the log-partition function is bounded above by twice the distance, in model parameter space, to a class of "standard" Ising models, for which variable inter-dependence is described via a simple mean field term. In the context of maximum-likelihood, using the new bound instead of the exact log-partition function, while constraining the distance to the class of standard Ising models, leads not only to a good approximation to the log-partition function, but also to a model that is parsimonious, and easily interpretable. We compare our bound with the log-determinant bound introduced by Wainwright and Jordan (2006), and show that when the $l_1$-norm of the model parameter vector is small enough, the latter is outperformed by the new bound.

## 1 Introduction

### 1.1 Problem statement

This paper is motivated by the problem fitting of binary distributions to experimental data. In the second-order Ising model, **PUT REF HERE** the fitted distribution $p$ is assumed to have the parametric form

$$p(x; Q, q) = \exp(x^T Q x + q^T x - Z(Q, q)), \ \ x \in \{0, 1\}^n,$$

where $Q = Q^T \in \mathbf{R}^n$ and $q \in \mathbf{R}^n$ contain the parameters of the model, and $Z(Q, q)$, the normalization constant, is called the *log-partition* function of the model. Noting that $x^T Q x + q^T x = x^T (Q + D(q)) x$ for every $x \in \{0, 1\}^n$, we will without loss of generality assume that $q = 0$, and denote by $Z(Q)$ the corresponding log-partition function

$$Z(Q) := \log \left( \sum_{x \in \{0,1\}^n} \exp[x^T Q x] \right). \tag{1}$$

In the Ising model, the maximum-likelihood approach to fitting data leads to the problem

$$\min_{Q \in \mathcal{Q}} \ Z(Q) - \mathbf{Tr} Q S, \tag{2}$$

where $\mathcal{Q}$ is a subset of the set $\mathcal{S}^n$ of symmetric matrices, and $S \in \mathcal{S}_+^n$ is the empirical second-moment matrix. When $\mathcal{Q} = \mathcal{S}^n$, the dual to (2) is the maximum entropy problem

$$\max_p \ H(p) \ : \ p \in \mathcal{P}, \ \ S = \sum_{x \in \{0,1\}^n} p(x) x x^T, \tag{3}$$

where $\mathcal{P}$ is the set of distributions with support in $\{0,1\}^n$, and $H$ is the entropy

$$H(p) = - \sum_{x \in \{0,1\}^n} p(x) \log p(x). \tag{4}$$

The constraints of problem (3) define a polytope in $\mathbf{R}^{2^n}$ called the marginal polytope.

For general $Q$'s, computing the log-partition function is NP-hard. Hence, except for special choices of $\mathcal{Q}$, the maximum-likelihood problem (2) is also NP-hard. It is thus desirable to find computationally tractable approximations to the log-partition function, such that the resulting maximum-likelihood problem is also tractable. In this regard, convex, upper bounds on the log-partition function are of particular interest, and our focus here: convexity usually brings about computational tractability, while using upper bounds yields a parameter $Q$ that is suboptimal for the exact problem.

Using an upper bound in lieu of $Z(Q)$ in (2), leads to a problem we will generically refer to as the *pseudo maximum-likelihood* problem. This corresponds to a relaxation to the maximum-entropy problem, which is (3) when $\mathcal{Q} = \mathcal{S}^n$. Such relaxations may involve two ingredients: an upper bound on the entropy, and an outer approximation to the marginal polytope.

## 1.2 Prior work

Due to the vast applicability of Ising models, the problem of approximating their log-partition function, and the related maximum-likelihood problem, has received considerable attention in the literature for decades, first in statistical physics, and more recently in machine learning.

The so-called log-determinant bound has been recently introduced, for a large class of Markov random fields, by Wainwright and Jordan [2]. (Their paper provides an excellent overview of the prior work, in the general context of graphical models.) The log-determinant bound is based on an upper bound on the differential entropy of continuous random variable, that is attained for a Gaussian distribution. The log-determinant bound enjoys good tractability properties, both for the computation of the log-partition function, and in the context of the maximum-likelihood problem (2). A recent paper by Ravikumar and Lafferty [1] discusses using bounds on the log-partition function to estimate marginal probabilities for a large class of graphical models, which adds extra motivation for the present study.

## 1.3 Main results and outline

The main purpose of this note is to introduce a new upper bound on the log-partition function that is computationally tractable. The new bound is convex in $Q$, and leads to a restriction to the maximum-likelihood problem that is also tractable. Our development crucially involves a specific class of Ising models, which we'll refer to as *standard Ising models*, in which the model parameter $Q$ has the form $Q = \mu I + \lambda \mathbf{1}\mathbf{1}^T$, where $\lambda, \mu$ are arbitrary scalars. Such models are indeed standard in statistical physics: the first term $\mu I$ describes interaction with the external magnetic field, and the second ($\lambda \mathbf{1}\mathbf{1}^T$) is a simple mean field approximation to ferro-magnetic coupling.

For standard Ising models, it can be shown that the log-partition functions has a computationally tractable, closed-form expression. Due to space limitation, such proof is omitted in this paper. Our bound is constructed so as to be exact in the case of standard Ising models. In fact, the error between our bound and the true value of the log-partition function is bounded above by twice the $l_1$-norm distance from the model parameters ($Q$) to the class of standard Ising models.

The outline of the note reflects our main results: in section 2, we introduce our bound, and show that the approximation error is bounded above by the distance to the class of standard Ising models. We discuss in section 3 the use of our bound in the context of the maximum-likelihood problem (2) and its dual (3). In particular, we discuss how imposing a bound on the distance to the class of standard Ising models may be desirable, not only to obtain an accurate approximation to the log-partition function, but also to find a parsimonious model, having good interpretability properties. We then compare the new bound with the log-determinant bound of Wainwright and Jordan in section 4. We show

that our new bound outperforms the log-determinant bound when the norm $\|Q\|_1$ is small enough (less than $0.08n$), and provide numerical experiments supporting the claim that our comparison analysis is quite conservative: our bound appears to be better over a wide range of values of $\|Q\|_1$.

**Notation.** Throughout the note, $n$ is a fixed integer. For $k \in \{0, \ldots, n\}$, define $\Delta_k := \{x \in \{0, 1\}^n : \textbf{Card}(x) = k\}$. Let $c_k = |\Delta_k|$ denote the cardinal of $\Delta_k$, and $\pi_k := 2^{-n} c_k$ the probability of $\Delta_k$ under the uniform distribution.

For a distribution $p$, the notation $\mathbf{E}_p$ refers to the corresponding expectation operator, and $\textbf{Prob}_p(S)$ to the probability of the event $S$ under $p$. The set $\mathcal{P}$ is the set of distributions with support on $\{0, 1\}^n$.

For $X \in \mathbf{R}^{n \times n}$, the notation $\|X\|_1$ denotes the sum of the absolute values of the elements of $X$, and $\|X\|_\infty$ the largest of these values. The set $\mathcal{S}^n$ is the set of symmetric matrices, $\mathcal{S}^n_+$ the set of symmetric positive semidefinite matrices. We use the notation $X \succeq 0$ for the statement $X \in \mathcal{S}^n_+$. If $x \in \mathbf{R}^n$, $D(x)$ is the diagonal matrix with $x$ on its diagonal. If $X \in \mathbf{R}^{n \times n}$, $d(X)$ is the $n$-vector formed with the diagonal elements of $X$. Finally, $\mathcal{X}$ is the set $\{(X, x) \in \mathcal{S}^n \times \mathbf{R}^n : d(X) = x\}$ and $\mathcal{X}_+ = \{(X, x) \in \mathcal{S}^n \times \mathbf{R}^n : X \succeq xx^T, d(X) = x\}$.

## 2 The Cardinality Bound

### 2.1 The maximum bound

To ease our derivation, we begin with a simple bound based on replacing each term in the log-partition function by its maximum over $\{0, 1\}^n$. This leads to an upper bound on the log-partition function:

$$Z(Q) \leq n \log 2 + \phi_{\max}(Q),$$

where

$$\phi_{\max}(Q) := \max_{x \in \{0,1\}^n} x^T Q x.$$

Computing the above quantity is in general NP-hard. Starting with the expression

$$\phi_{\max}(Q) = \max_{(X,x) \in \mathcal{X}_+} \mathbf{Tr} Q X : \textbf{rank}(X) = 1,$$

and relaxing the rank constraint leads to the upper bound $\phi_{\max}(Q) \leq \psi_{\max}(Q)$, where $\psi_{\max}(Q)$ is defined via a semidefinite program:

$$\psi_{\max}(Q) = \max_{(X,x) \in \mathcal{X}_+} \mathbf{Tr} Q X, \tag{5}$$

where $\mathcal{X}_+ = \{(X, x) \in \mathcal{S}^n \times \mathbf{R}^n : X \succeq xx^T, d(X) = x\}$. For later reference, we note the dual form:

$$\psi_{\max}(Q) = \min_{t,\nu} t : \begin{pmatrix} D(\nu) - Q & \frac{1}{2}\nu \\ \frac{1}{2}\nu^T & t \end{pmatrix} \succeq 0 \tag{6}$$

$$= \min_\nu \frac{1}{4}\nu^T (D(\nu) - Q)^{-1}\nu : D(\nu) \succ Q. \tag{7}$$

The corresponding bound on the log-partition function, referred to as the *maximum* bound, is

$$Z(Q) \leq Z_{\max}(Q) := n \log 2 + \psi_{\max}(Q).$$

The complexity of this bound (using interior-point methods) is roughly $O(n^3)$.

Let us make a few observations before proceeding. First, the maximum-bound is a convex function of $Q$, which is important in the context of the maximum-likelihood problem (2). Second, we have $Z_{\max}(Q) \leq n \log 2 + \|Q\|_1$, which follows from (5), together with the fact that any matrix $X$ that is feasible for that problem satisfies $\|X\|_\infty \leq 1$. Finally, we observe that the function $Z_{\max}$ is Lipschitz continuous, with constant 1 with respect to the $l_1$-norm. It can be shown that the same property holds for the log-partition function $Z$ itself. Due to space limitation such proof is omitted in this paper. Indeed, for every symmetric matrices $Q, R$ we have the sub-gradient inequality

$$Z_{\max}(R) \geq Z_{\max}(Q) + \mathbf{Tr} X^{\text{opt}}(R - Q),$$

where $X^{\text{opt}}$ is any optimal variable for the dual problem (5). Since any feasible $X$ satisfies $\|X\|_\infty \leq 1$, we can bound the term $\mathbf{Tr} X^{\text{opt}}(Q - R)$ from below by $-\|Q - R\|_1$, and after exchanging the roles of $Q, R$, obtain the desired result.

## 2.2 The cardinality bound

For every $k \in \{0, \ldots, n\}$, consider the subset of variables with cardinality $k$, $\Delta_k := \{x \in \{0,1\}^n : \mathbf{Card}(x) = k\}$. This defines a partition of $\{0,1\}^n$, thus

$$Z(Q) = \log \left( \sum_{k=0}^{n} \sum_{x \in \Delta_k} \exp[x^T Q x] \right).$$

We can refine the maximum bound by replacing the terms in the log-partition by their maximum over $\Delta_k$, leading to

$$Z(Q) \le \log \left( \sum_{k=0}^{n} c_k \exp[\phi_k(Q)] \right),$$

where, for $k \in \{0, \ldots, n\}$, $c_k = |\Delta_k|$, and

$$\phi_k(Q) := \max_{x \in \Delta_k} x^T Q x.$$

Computing $\phi_k(Q)$ for arbitrary $k \in \{0, \ldots, n\}$ is NP-hard. Based on the identity

$$\phi_k(Q) = \max_{(X,x) \in \mathcal{X}_+} \mathbf{Tr} QX \ : \ x^T x = k, \ \mathbf{1}^T X \mathbf{1} = k^2, \ \mathbf{rank} X = 1, \tag{8}$$

and using rank relaxation as before, we obtain the bound $\phi_k(Q) \le \psi_k(Q)$, where

$$\psi_k(Q) = \max_{(X,x) \in \mathcal{X}_+} \mathbf{Tr} QX \ : \ x^T x = k, \ \mathbf{1}^T X \mathbf{1} = k^2. \tag{9}$$

We define the *cardinality bound,* as

$$Z_{\mathrm{card}}(Q) := \log \left( \sum_{k=0}^{n} c_k \exp[\psi_k(Q)] \right).$$

The complexity of computing $\psi_k(Q)$ (using interior-point methods) is roughly $O(n^3)$. The upper bound $Z_{\mathrm{card}}(Q)$ is computed via $n$ semidefinite programs of the form (9). Hence, its complexity is roughly $O(n^4)$.

Problem (9) admits the dual form

$$\psi_k(Q) \quad := \quad \min_{t,\mu,\nu,\lambda} \ t + k\mu + \lambda k^2 \ : \ \begin{pmatrix} D(\nu) + \mu I + \lambda \mathbf{1}\mathbf{1}^T - Q & \frac{1}{2}\nu \\ \frac{1}{2}\nu^T & t \end{pmatrix} \succeq 0. \tag{10}$$

The fact that $\psi_k(Q) \le \psi_{\max}(Q)$ for every $k$ is obtained upon setting $\lambda = \mu = 0$ in the semi-definite programming problem (10). In fact, we have

$$\psi_k(Q) = \min_{\mu,\lambda} \ k\mu + k^2\lambda + \psi_{\max}(Q - \mu I - \lambda \mathbf{1}\mathbf{1}^T). \tag{11}$$

The above expression can be directly obtained from the following, valid for every $\mu, \lambda$:

$$\begin{aligned} \phi_k(Q) &= k\mu + k^2\lambda + \phi_k(Q - \mu I - \lambda \mathbf{1}\mathbf{1}^T) \\ &\le k\mu + k^2\lambda + \phi_{\max}(Q - \mu I - \lambda \mathbf{1}\mathbf{1}^T) \\ &\le k\mu + k^2\lambda + \psi_{\max}(Q - \mu I - \lambda \mathbf{1}\mathbf{1}^T). \end{aligned}$$

It can be shown (proof which we omit due to space limitation) that, in the case of standard Ising models, that is if $Q$ has the form $\mu I + \lambda \mathbf{1}\mathbf{1}^T$ for some scalars $\mu, \lambda$, then the bound $\psi_k(Q)$ is exact. Since the values of $x^T Q x$ when $x$ ranges $\Delta_k$ are constant, the cardinality bound is also exact.

By construction, $Z_{\mathrm{card}}(Q)$ is guaranteed to be better (lower) than $Z_{\max}(Q)$, since the latter is obtained upon replacing $\psi_k(Q)$ by its upper bound $\psi(Q)$ for every $k$. The cardinality bound thus satisfies

$$Z(Q) \le Z_{\mathrm{card}}(Q) \le Z_{\max}(Q) \le n \log 2 + \|Q\|_1. \tag{12}$$

Using the same technique as used in the context of the maximum bound, we can show that the function $\psi_k$ is Lipschitz-continuous, with constant 1 with respect to the $l_1$-norm. Using the Lipschitz continuity of positively weighted log-sum-exp functions (with constant 1 with respect to the $l_\infty$ norm), we deduce that $Z_{\mathrm{card}}(Q)$ is also Lipschitz-continuous: for every symmetric matrices $Q, R$,

$$
\begin{aligned}
|Z_{\mathrm{card}}(Q) - Z_{\mathrm{card}}(R)| &\leq \left| \log\left( \sum_{k=0}^{n} c_k \exp[\psi_k(Q)] \right) - \log\left( \sum_{k=0}^{n} c_k \exp[\psi_k(R)] \right) \right| \\
&\leq \max_{0 \leq k \leq n} |\psi_k(Q) - \psi_k(R)| \\
&\leq \|Q - R\|_1,
\end{aligned}
$$

as claimed.

### 2.3 Quality analysis

We now seek to establish conditions on the model parameter $Q$, which guarantee that the approximation error $Z_{\mathrm{card}}(Q) - Z(Q)$ is small. The analysis relies on the fact that, for standard Ising models, the error is zero.

We begin by establishing an upper bound on the difference between maximal and minimal values of $x^T Q x$ when $x \in \Delta_k$. We have the bound

$$
\min_{x \in \Delta_k} x^T Q x \geq \eta_k(Q) := \min_{(X,x) \in \mathcal{X}_+} \mathbf{Tr} QX \; : \; x^T x = k, \; \mathbf{1}^T X \mathbf{1} = k^2.
$$

In the same fashion as for the quantity $\psi_k(Q)$, we can express $\eta_k(Q)$ as

$$
\eta_k(Q) = \max_{\mu, \lambda} k\mu + k^2 \lambda + \psi_{\min}(Q - \mu I - \lambda \mathbf{1}\mathbf{1}^T),
$$

where $\psi_{\min}(Q) := \min_{(X,x) \in \mathcal{X}_+} \mathbf{Tr} QX$. Based on this expression, we have, for every $k$:

$$
\begin{aligned}
0 \leq \psi_k(Q) - \eta_k(Q) \;=\; \min_{\lambda,\mu,\,\lambda',\mu'} \quad & k(\mu - \mu') + k^2(\lambda - \lambda') + \\
& \psi_{\max}(Q - \mu I - \lambda\mathbf{1}\mathbf{1}^T) - \psi_{\min}(Q - \mu' I - \lambda'\mathbf{1}\mathbf{1}^T) \\
\leq \min_{\lambda,\mu} \quad & \psi_{\max}(Q - \mu I - \lambda\mathbf{1}\mathbf{1}^T) - \psi_{\min}(Q - \mu I - \lambda\mathbf{1}\mathbf{1}^T) \\
\leq 2\min_{\lambda,\mu} \quad & \|Q - \mu I - \lambda\mathbf{1}\mathbf{1}^T\|_1,
\end{aligned}
$$

where we have used the fact that, for every symmetric matrix $R$, we have

$$
\begin{aligned}
0 \leq \psi_{\max}(R) - \psi_{\min}(R) &= \max_{(X,x),(Y,y) \in \mathcal{X}_+} \mathbf{Tr} R(X - Y) \\
&\leq \max_{\|X\|_\infty \leq 1, \|Y\|_\infty \leq 1} \mathbf{Tr} R(X - Y) \\
&= 2\|R\|_1.
\end{aligned}
$$

Using again the Lipschitz continuity properties of the weighted log-sum-exp function, we obtain that for every $Q$, the absolute error between $Z(Q)$ and $Z_{\mathrm{card}}(Q)$ is bounded as follows:

$$
\begin{aligned}
0 \leq Z_{\mathrm{card}}(Q) - Z(Q) &\leq \log\left( \sum_{k=0}^{n} c_k \exp[\psi_k(Q)] \right) - \log\left( \sum_{k=0}^{n} c_k \exp[\eta_k(Q)] \right) \\
&\leq \max_{0 \leq k \leq n} (\psi_k(Q) - \eta_k(Q)) \\
&\leq 2D_{\mathrm{st}}(Q), \quad D_{\mathrm{st}}(Q) := \min_{\lambda,\mu} \|Q - \mu I - \lambda\mathbf{1}\mathbf{1}^T\|_1, \quad (13)
\end{aligned}
$$

Thus, a measure of quality is $D_{\mathrm{st}}(Q)$, the distance, in $l_1$-norm, between the model and the class of standard Ising models. Note that this measure is easily computed, in $O(n^2 \log n)$ time, by first setting $\lambda$ to be the median of the values $Q_{ij}$, $1 \leq i < j \leq n$, and then setting $\mu$ to be the median of the values $Q_{ii} - \lambda$, $i = 1, \ldots, n$.

We summarize our findings so far with the following theorem:

**Theorem 1 (Cardinality bound)** *The cardinality bound is*

$$Z_{\text{card}}(Q) := \log \left( \sum_{k=0}^{n} c_k \exp[\psi_k(Q)] \right).$$

*where $\phi_k(Q)$, $k = 0, \ldots, n$, is defined via the semidefinite program (9), which can be solved in $O(n^3)$. The approximation error is bounded above by twice the distance (in $l_1$-norm) to the class of standard Ising models:*

$$0 \leq Z_{\text{card}}(Q) - Z(Q) \leq 2 \min_{\lambda, \mu} \|Q - \mu I - \lambda \mathbf{1}\mathbf{1}^T\|_1.$$

## 3  The Pseudo Maximum-Likelihood Problem

### 3.1  Tractable formulation

Using the bound $Z_{\text{card}}(Q)$ in lieu of $Z(Q)$ in the maximum-likelihood problem (2) leads to a convex restriction of that problem, referred to as the pseudo-maximum likelihood problem. This problem can be cast as

$$\min_{t, \mu, \nu, Q} \quad \log \left( \sum_{k=0}^{n} c_k \exp[t_k + k\mu_k + k^2 \lambda_k] \right) - \mathbf{Tr}QS$$

$$\text{s.t. } Q \in \mathcal{Q}, \quad \begin{pmatrix} D(\nu_k) + \mu_k I + \lambda_k \mathbf{1}\mathbf{1}^T - Q & \frac{1}{2}\nu_k \\ \frac{1}{2}\nu_k^T & t_k \end{pmatrix} \succeq 0, \quad k = 0, \ldots, n.$$

The complexity of this bound is . For numerical reasons, and without loss of generality, it is advisable to scale the $c_k$'s and replace them by $\pi_k := 2^{-n} c_k \in [0, 1]$.

### 3.2  Dual and interpretation

When $\mathcal{Q} = \mathcal{S}^n$, the dual to the above problem is

$$\max_{(Y_k, y_k, q_k)_{k=0}^{n}} -D(q\|\pi) \quad : \quad S = \sum_{k=0}^{n} Y_k, \quad q \geq 0, \quad q^T \mathbf{1} = 1,$$

$$\begin{pmatrix} Y_k & y_k \\ y_k^T & q_k \end{pmatrix} \succeq 0, \quad d(Y_k) = y_k,$$

$$\mathbf{1}^T y_k = k q_k, \quad \mathbf{1}^T Y_k \mathbf{1} = k^2 q_k, \quad k = 0 \ldots, n.$$

where $\pi$ is the distribution on $\{0, \ldots, n\}$, with $\pi_k = \mathbf{Prob}_u \Delta_k = 2^{-n} c_k$, and $D(q\|\pi)$ is the relative entropy (Kullback-Leibler divergence) between the distributions $q, \pi$:

$$D(q\|\pi) := \sum_{k=0}^{n} q_k \log \frac{q_k}{\pi_k}.$$

To interpret this dual, we assume without loss of generality $q > 0$, and use the variables $X_k := q_k^{-1} Y_k$, $x_k := q_k^{-1} y_k$. We obtain the equivalent (non-convex) formulation

$$\max_{(X_k, x_k, q_k)_{k=0}^{n}} -D(q\|\pi) \quad : \quad S = \sum_{k=0}^{n} q_k X_k, \quad q \geq 0, \quad q^T \mathbf{1} = 1, \tag{14}$$

$$(X_k, x_k) \in \mathcal{X}_+, \quad \mathbf{1}^T x_k = k, \quad \mathbf{1}^T X_k \mathbf{1} = k^2, \quad k = 0 \ldots, n.$$

The above problem can be obtained as a relaxation to the dual of the exact maximum-likelihood problem (2), which is the maximum entropy problem (3). The relaxation involves two steps: one is to form an outer approximation to the marginal polytope, the other is to find an upper bound on the entropy function (4).

First observe that we can express any distribution on $\{0,1\}^n$ as

$$p(x) = \sum_{k=0}^{n} q_k p_k(x), \tag{15}$$

where

$$q_k = \mathbf{Prob}_p \Delta_k = \sum_{x \in \Delta_k} p(x), \quad p_k(x) = \begin{cases} q_k^{-1} p(x) & \text{if } x \in \Delta_k, \\ 0 & \text{otherwise.} \end{cases}$$

Note that the functions $p_k$ are valid distributions on $\{0,1\}^n$ as well as $\Delta_k$.

To obtain an outer approximation to the marginal polytope, we then write the moment-matching equality constraint in problem (3) as

$$S = \mathbf{E}_p x x^T = \sum_{k=0}^{n} q_k X_k,$$

where $X_k$'s are the second-order moment matrices with respect to $p_k$:

$$X_k = \mathbf{E}_{p_k} x x^T = q_k^{-1} \sum_{x \in \Delta_k} p(x) x x^T.$$

To relax the constraints in the maximum-entropy problem (3), we simply use the valid constraints $X_k \succeq x_k x_k^T$, $d(X_k) = x_k$, $\mathbf{1}^T x_k = k$, $\mathbf{1}^T X_k \mathbf{1} = k^2$, where $x_k$ is the mean under $p_k$:

$$x_k = \mathbf{E}_{p_k} x = q_k^{-1} \sum_{x \in \Delta_k} p(x) x.$$

This process yields exactly the constraints of the relaxed problem (14).

To finalize our relaxation, we now form an upper bound on the entropy function (4). To this end, we use the fact that, since each $p_k$ has support in $\Delta_k$, its entropy is bounded above by $\log |\Delta_k|$, as follows:

$$
\begin{aligned}
-H(p) = \sum_{x \in \{0,1\}^n} p(x) \log p(x) &= \sum_{k=0}^{n} \sum_{x \in \Delta_k} p(x) \log p(x) \\
&= \sum_{k=0}^{n} \sum_{x \in \Delta_k} q_k p_k(x) \log(q_k p_k(x)) \\
&= \sum_{k=0}^{n} q_k (\log q_k - H(p_k)) \\
&\geq \sum_{k=0}^{n} q_k (\log q_k - \log |\Delta_k|) \quad (|\Delta_k| = 2^n \pi_k) \\
&\geq \sum_{k=0}^{n} q_k \log \frac{q_k}{\pi_k} - n \log 2,
\end{aligned}
$$

which is, up to a constant, the objective of problem (14).

### 3.3 Ensuring quality via bounds on $Q$

We consider the (exact) maximum-likelihood problem (2), with $\mathcal{Q} = \{Q = Q^T : \|Q\|_1 \leq \epsilon\}$:

$$\min_{Q=Q^T} Z(Q) - \mathbf{Tr}QS : \|Q\|_1 \leq \epsilon, \tag{16}$$

and its convex relaxation:

$$\min_{Q=Q^T} Z_{\mathrm{card}}(Q) - \mathbf{Tr}QS : \|Q\|_1 \leq \epsilon. \tag{17}$$

The feasible sets of problems (16) and (17) are the same, and on it the difference in the objective functions is uniformly bounded by $2\epsilon$. Thus, any $\epsilon$-suboptimal solution of the relaxation (17) is guaranteed to by $3\epsilon$-suboptimal for the exact problem, (16).

In practice, the $l_1$-norm constraint in (17) encourages sparsity of $Q$, hence the interpretability of the model. It also has good properties in terms of the generalization error. As seen above, the constraint also implies a better approximation to the exact problem (16). All these benefits come at the expense of goodness-of-fit, as the constraint reduces the expressive power of the model. This is an illustration of the intimate connections between computational and statistical properties of the model.

A more accurate bound on the approximation error can be obtained by imposing the following constraint on $Q$ and two new variables $\lambda, \mu$:

$$\|Q - \mu I - \lambda \mathbf{1} \mathbf{1}^T\|_1 \leq \epsilon.$$

We can draw similar conclusions as before. Here, the resulting model will not be sparse, in the sense of having many elements in $Q$ equal to zero. However, it will still be quite interpretable, as the bound above will encourage the number of off-diagonal elements in $Q$ that differ from their median, to be small.

A yet more accurate control on the approximation error can be induced by the constraints $\psi_k(Q) \leq \epsilon + \eta_k(Q)$ for every $k$, each of which can be expressed as an LMI constraint. The corresponding constrained relaxation to the maximum-likelihood problem has the form

$$\min_{t, \mu^{\pm}, \nu^{\pm}, Q} \quad \log\left(\sum_{k=0}^n c_k \exp[t_k^+ + k\mu_k^+ + k^2\lambda_k^+]\right) - \mathbf{Tr}QS$$

$$\text{s.t.} \quad \begin{pmatrix} \mathbf{diag}(\nu_k^+) + \mu_k^+ I + \lambda_k^+ \mathbf{1}\mathbf{1}^T - Q & \frac{1}{2}\nu_k^+ \\ \frac{1}{2}\nu_k^+ & t_k^+ \end{pmatrix} \succeq 0, \quad k = 0, \ldots, n,$$

$$\begin{pmatrix} Q - \mathbf{diag}(\nu_k^-) - \mu_k^- I - \lambda_k^- \mathbf{1}\mathbf{1}^T & \frac{1}{2}\nu_k^- \\ \frac{1}{2}\nu_k^- & t_k^- \end{pmatrix} \succeq 0, \quad k = 0, \ldots, n,$$

$$t_k^+ - t_k^- \leq \epsilon, \quad k = 0, \ldots, n.$$

Using this model instead of ones we saw previously, we sacrifice less on the front of the approximation to the true likelihood, at the expense of increased computational effort.

## 4 Links with the Log-Determinant Bound

### 4.1 The log-determinant bounds

The bound in Wainwright and Jordan [2] is based on an upper bound on the (differential) entropy of a continuous random variable, which is attained for a Gaussian distribution. It has the form $Z(Q) \leq Z_{\mathrm{ld}}(Q)$, with

$$Z_{\mathrm{ld}}(Q) := \alpha n + \max_{(X,x)\in\mathcal{X}_+} \mathbf{Tr}QX + \frac{1}{2}\log\det(X - xx^T + \frac{1}{12}I) \tag{18}$$

where $\alpha := (1/2)\log(2\pi e) \approx 1.42$. Wainwright and Jordan suggest to further relax this bound to one which is easier to compute:

$$Z_{\mathrm{ld}}(Q) \leq Z_{\mathrm{rld}}(Q) := \alpha n + \max_{(X,x)\in\mathcal{X}} \mathbf{Tr}QX + \frac{1}{2}\log\det(X - xx^T + \frac{1}{12}I). \tag{19}$$

Like $Z$ and the bounds examined previously, the bound $Z_{\mathrm{ld}}$ and $Z_{\mathrm{rld}}$ are Lipschitz-continuous, with constant 1 with respect to the $l_1$ norm. The proof starts with the representations above, and exploits the fact that $\|Q\|_1$ is an upper bound on $\mathbf{Tr}QX$ when $(X,x) \in \mathcal{X}_+$.

The dual of the log-determinant bound has the form (see appendix (**??**))

$$Z_{\mathrm{ld}}(Q) = \frac{n}{2}\log \pi - \frac{1}{2}\log 2 +$$

$$\min_{t,\nu,F,g,h} \; t + \frac{1}{12}\mathbf{Tr}(D(\nu) - Q - F) - \frac{1}{2}\log\det \begin{pmatrix} D(\nu) - Q - F & -\frac{1}{2}\nu - g \\ -\frac{1}{2}\nu^T - g^T & t - h \end{pmatrix}$$

$$\text{s.t.} \; \begin{pmatrix} F & g \\ g & h \end{pmatrix} \succeq 0. \tag{20}$$

The relaxed counterpart $Z_{\mathrm{rld}}(Q)$ is obtained upon setting $F, g, h$ to zero in the dual above:

$$Z_{\mathrm{rld}}(Q) = \frac{n}{2}\log\pi - \frac{1}{2}\log 2 + \min_{t,\nu} \; t + \frac{1}{12}\mathbf{Tr}(D(\nu) - Q) - \frac{1}{2}\log\det\begin{pmatrix} D(\nu) - Q & -\frac{1}{2}\nu \\ -\frac{1}{2}\nu^T & t \end{pmatrix}.$$

Using Schur complements to eliminate the variable $t$, we further obtain

$$Z_{\mathrm{rld}}(Q) = \frac{n}{2}\log\pi + \frac{1}{2} +$$

$$\min_{\nu} \; \frac{1}{4}\nu^T(D(\nu) - Q)^{-1}\nu + \frac{1}{12}\mathbf{Tr}(D(\nu) - Q) - \frac{1}{2}\log\det(D(\nu) - Q). \tag{21}$$

## 4.2 Comparison with the maximum bound

We first note the similarity in structure between the dual problem (5) defining $Z_{\max}(Q)$ and that of the relaxed log-determinant bound.

Despite these connections, the log-determinant bound is neither better nor worse than the cardinality or maximum bounds. Actually, for some special choices of $Q$ (e.g. when $Q$ is diagonal), the cardinality bound is exact, while the log-determinant one is not. Conversely, one can choose $Q$ so that $Z_{\mathrm{card}}(Q) > Z_{\mathrm{ld}}(Q)$, so no bound dominates the other. The same can be said for $Z_{\max}(Q)$ (see section 4.4 for numerical examples).

However, when we impose an extra condition on $Q$, namely a bound on its $l_1$ norm, more can be said. The analysis is based on the case $Q = 0$, and exploits the Lipschitz continuity of the bounds with respect to the $l_1$-norm.

First notice (although not shown in this paper because of space limitation) that, for $Q = 0$, the relaxed log-determinant bound writes

$$Z_{\mathrm{rld}}(0) = \frac{n}{2}\log\frac{2\pi e}{3} + \frac{1}{2}$$

$$= Z_{\max}(0) + \frac{n}{2}\log\frac{\pi e}{6} + \frac{1}{2}.$$

Now invoke the Lipschitz continuity properties of the bounds $Z_{\mathrm{rld}}(Q)$ and $Z_{\max}(Q)$, and obtain that

$$Z_{\mathrm{rld}}(Q) - Z_{\max}(Q) = (Z_{\mathrm{rld}}(Q) - Z_{\mathrm{rld}}(0)) + (Z_{\mathrm{rld}}(0) - Z_{\max}(0)) + (Z_{\max}(0) - Z_{\max}(Q))$$

$$\geq -2\|Q\|_1 + (Z_{\mathrm{rld}}(0) - Z_{\max}(0))$$

$$= -2\|Q\|_1 + + \frac{n}{2}\log\frac{\pi e}{6} + \frac{1}{2}.$$

This proves that if $\|Q\|_1 \leq \frac{n}{4}\log\frac{\pi e}{6} + \frac{1}{4}$, then the relaxed log-determinant bound $Z_{\mathrm{rld}}(Q)$ is worse (larger) than the maximum bound $Z_{\max}(Q)$. We can strengthen the above condition to $\|Q\|_1 \leq 0.08n$.

## 4.3 Summary of comparison results

To summarize our findings:

**Theorem 2 (Comparison)** *We have for every $Q$:*

$$Z(Q) \leq Z_{\mathrm{card}}(Q) \leq Z_{\max}(Q) \leq n\log 2 + \|Q\|_1.$$

*In addition, we have $Z_{\max}(Q) \leq Z_{\mathrm{rld}}(Q)$ whenever $\|Q\|_1 \leq 0.08n$.*

### 4.4 A numerical experiment

We now illustrate our findings on the comparison between the log-determinant bounds and the cardinality and maximum bounds. We set the size of our model to be $n = 20$, and for a range of values of a parameter $\rho$, generate $N = 10$ random instances of $Q$ with $\|Q\|_1 = \rho$. Figure **??** shows the average values of the bounds, as well as the associated error bars. Clearly, the new bound outperforms the log-determinant bounds for a wide range of values of $\rho$. Our predicted threshold value of $\|Q\|_1$ for which the new bound becomes worse, namely $\rho = 0.08n \approx 1.6$ is seen to be very conservative, with respect to the observed threshold of $\rho \approx 30$. On the other hand, we observe that for large values of $\|Q\|_1$, the log-determinant bounds do behave better. Across the range of $\rho$, we note that the log-determinant bound is indistinguishable from its relaxed counterpart.

## 5 Conclusion and Remarks

We have introduced a new upper bound (*the cardinality bound*) for the log-partition function corresponding to second-order Ising models for binary distribution. We have shown that such a bound can be computed via convex optimization, and, when compared to the log-determinant bound introduced by Wainwright and Jordan (2006), the cardinality bound performs better when the $l_1$-norm of the model parameter vector is small enough.

Although not shown in the paper, the cardinality bound becomes exact in the case of standard Ising model, while the maximum bound (for example) is not exact for such model.

As was shown in section 2, the cardinality bound was computed by defining a partition of $\{0, 1\}$. This idea can be generalized to form a class of bounds which we call *partition bounds*. It turns out that partitions bound are closely linked to the more general class bounds that are based on worst-case probability analysis.

We acknowledge the importance of applying our bound to real-word data. We hope to include such results in subsequent versions of this paper.

## References

[1] P. Ravikumar and J. Lafferty. Variational Chernoff bounds for graphical models. In *Proc. Advances in Neural Information Processing Systems (NIPS)*, December 2007.

[2] Martin J. Wainwright and Michael I. Jordan. Log-determinant relaxation for approximate inference in discrete Markov random fields. *IEEE Trans. Signal Processing*, 2006.
